# Coupled Markov Random Fields and Mean Field Theory

**Davi Geiger**[1]
Artificial Intelligence
Laboratory, MIT
545 Tech. Sq. # 792
Cambridge, MA 02139

and

**Federico Girosi**
Artificial Intelligence
Laboratory, MIT
545 Tech. Sq. # 788
Cambridge, MA 02139

## ABSTRACT

In recent years many researchers have investigated the use of Markov Random Fields (MRFs) for computer vision. They can be applied for example to reconstruct surfaces from sparse and noisy depth data coming from the output of a visual process, or to integrate early vision processes to label physical discontinuities. In this paper we show that by applying mean field theory to those MRFs models a class of neural networks is obtained. Those networks can speed up the solution for the MRFs models. The method is not restricted to computer vision.

## 1    Introduction

In recent years many researchers (Geman and Geman, 1984) (Marroquin et. al. 1987) (Gamble et. al. 1989) have investigated the use of Markov Random Fields (MRFs) for early vision. Coupled MRFs models can be used for the reconstruction of a function starting from a set of noisy sparse data, such as intensity, stereo, or motion data. They have also been used to integrate early vision processes to label physical discontinuities. Two fields are usually required in the MRFs formulation of a problem: one represents the function that has to be reconstructed, and the other is associated to its discontinuities. The reconstructed function, say $f$, has

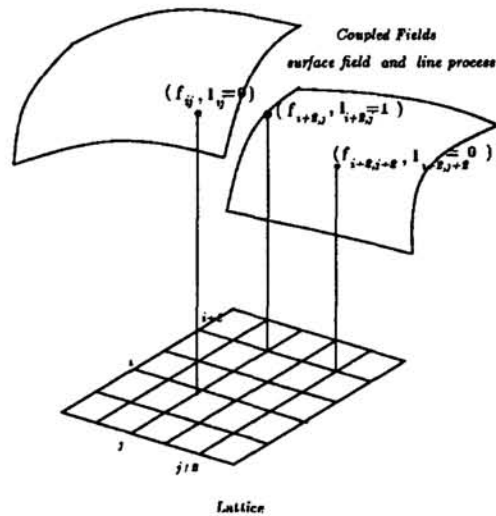

**Figure 1:** *The square lattice with the line process l and the field f defined at some pixels.*

a continuous range and the discontinuity field, say $l$, is a binary field (1 if there is a discontinuity and 0 otherwise, see figure 1). The essence of the MRFs model is that the probability distribution of the configuration of the fields, for a given a set of data, has a Gibbs distribution for some cost functional dependent upon a small neighborhood. Since the fields have a discrete range, to find the solution becomes a combinatorial optimization problem, that can be solved by means of methods like the Monte Carlo one (simulated annealing (Kirkpatrick and all, 1983), for example). However it has a main drawback: the amount of computer time needed for the implementation.

We propose to approximate the solution of the problem formulated in the MRFs frame with its "average solution." The mean field theory (MFT) allows us to find deterministic equations for MRFs whose solution approximates the solution of the statistical problem. A class of neural networks can naturally solve these equations (Hopfield, 1984) (Koch et. al., 1985) (Geiger and Yuille, 1989). An advantage of such an approach is that the solution of the networks is faster than the Monte Carlo techniques, commonly used to deal with MRFs.

A main novelty in this work, and a quite general one, is to show that the binary field representing the discontinuities can be averaged out to yield an effective theory independent of the binary field. The possibility of writing a set of equations describing the network is also useful for a better understanding of the nature of the solution and of the parameters of the model. We show the network performance in an example of image reconstruction from sparse data.

## 2    MRFs and Bayes approach

One of the main attractions of MRFs models in vision is that they can deal directly with discontinuities. We consider coupled MRFs depending upon two fields, $f$ and $l$. For the problem of image reconstruction the field $f$ represents the field to be smoothed and $l$ represents the discontinuities. In this case $l$ is a binary field, assuming the values 1 if there is a discontinuity and 0 otherwise. The Markov property asserts that the probability of a certain value of the field at any given site in the lattice depends only upon neighboring sites. According to the Clifford-Hammersley theorem, the prior probability of a state of the fields $f$ and $l$ has the Gibbs form:

$$P(f, l) = \frac{1}{Z} e^{-\beta U(f,l)} \qquad (2.1)$$

where $f$ and $l$ are the fields, e.g. the surface-field and its discontinuities, $Z$ is the normalization constant also known as the partition function, $U(f, l) = \sum_i U_i(f, l)$ is an energy function that can be computed as the sum of local contributions from each lattice site $i$, and $\beta$ is a parameter that is called the inverse of the natural temperature of the field. If a sparse observation $g$ for any given surface-field $f$ is given and a model of the noise is available then one knows the conditional probability $P(g|f, l)$. Bayes theorem then allows us to write the posterior distribution:

$$P(f, l|g) = \frac{P(g|f,l)P(f,l)}{P(g)} \equiv \frac{1}{Z} e^{-\beta V(f|g)} . \qquad (2.2)$$

For the case of a sparse image corrupted by white gaussian noise

$$V(f, l|g) = \sum_i \lambda_i (f_i - g_i)^2 + U_i(f, l) \qquad (2.3)$$

where $\lambda_{ij} = 1$ or $0$ depending on whether data are available or not. $V(f, l|g)$ is sometimes called the visual cost function. The solution for the problem is the given by some estimate of the fields. The maximum of the posterior distribution or other related estimates of the "true" data-field value can not be computed analytically, but sample distributions of the field with the probability distribution of (2.2) can be obtained using Monte Carlo techniques such as the Metropolis algorithm. These algorithms sample the space of possible values of the fields according to the probability distribution $P(f, l|g)$.

A drawback of coupled MRFs has been the amount of computer time used in the Metropolis algorithm or in simulated annealing (Kirkpatrick et. al., 1983).

A justification for using the mean field (MF) as a measure of the fields, $f$ for example, resides in the fact that it represents the minimum variance Bayes estimator. More precisely, the average variance of the field $f$ is given by

$$Var_f = \sum_{f,l}(f - \bar{f})^2 P(f,l|g)$$

where $\bar{f}$ is a given estimate of the field, the $\sum_{f,l}$ represents the sum over all the possible configurations of $f$ and $l$, and $Var_f$ is the variance. Minimizing $Var_f$ with respect to all possible values of $\bar{f}$ we obtain

$$\frac{\partial}{\partial \bar{f}}Var_f = 0 \Rightarrow \bar{f} = \sum_{f,l} fP(f,l|g)$$

This equation for $\bar{f}$ defines the deterministic MF equations.

## 2.1   MFT and Neural Networks

To connect MRFs to neural networks, we use Mean field theory (MFT) to obtain deterministic equations from MRFs that represent a class of neural networks.

The mean field for the values f and l at site $i$ are given by

$$\bar{f}_i = \sum_{f,l} f_i P(f,l|g) \quad \text{and} \quad \bar{l}_i = \sum_{f,l} l_i P(f,l|g) \tag{2.4}$$

The sum over the binary process, $l_i = 0,1$ gives for (2.3), using the mean field approximation,

$$\bar{f}_i = \sum_f f_i \frac{e^{-\beta\lambda_i(f_i-g_i)^2}}{Z_i}\left(e^{-\beta U_i(f,l_{j\neq i},l_i=0)} + e^{-\beta U_i(f,l_{j\neq i},l_i=1)}\right)$$

$$\bar{l}_i = \sum_f \frac{e^{-\beta[\lambda_i(f_i-g_i)^2+U_i(f,l_{j\neq i},l_i=1)]}}{Z_i} \tag{2.5}$$

where the partition function $Z$ where factorized as $\prod_i Z_i$. In this case

$$Z_i = \sum_f e^{-\beta\lambda_i(f_i-g_i)^2}\left(e^{-\beta U_i(f,l_{j\neq i},l_i=0)} + e^{-\beta U_i(f,l_{j\neq i},l_i=1)}\right).$$

Another way to write the equation for $f$ is

$$\bar{f}_i = \sum_f f_i \frac{e^{-\beta V_i^{effective}}}{Z_i} \tag{2.6}$$

where

$$V_i^{effective}(f) = \lambda_i(f_i - g_i)^2 - \frac{1}{\beta}ln(e^{-\beta U_i(f,l_{j\neq i},l_i=0)} + e^{-\beta U_i(f,l_{j\neq i},l_i=1)}) \quad (2.7)$$

The important result obtained here is that the effective potential does not dependend on the binary field $l_i$. The line process field has been eliminated to yield a temperature dependent effective potential (also called visual cost function). The interaction of the field $f$ with itself has changed after the line process has been averaged out. We interpret this result as the effect of the interaction of the line processes with the field $f$ to yield a new temperature dependent potential.

The computation of the sum over all the configurations of the field $f$ is hard and we use the saddle point approximation. In this case is equivalent to minimize $V^{effective}(f)$. A dynamical equation to find the minimum of $V^{effective}$ is given by introducing a damping force $\frac{\partial f}{\partial t}$ that brings the system to equilibrium. Therefore the mean field equation under the mean field and saddle point approximation becomes

$$\frac{\partial}{\partial f_i}V_i^{effective}(f,\bar{l}) = \frac{\partial f_i}{\partial t} \quad (2.8)$$

Equation (2.8) represents a class of unsupervised neural networks coupled to (2.5). The mean field solution is given by the fixed point of (2.8) and (2.5) it is attained after running (2.8) and (2.5) as $t \mapsto \infty$. This network is better understood with an example of image reconstruction.

## 3   Example: Image reconstruction

To reconstruct images from sparse data and to detect discontinuities we use the weak membrane model where $U_i(f,l)$ in two dimensions is given by

$$U_{i,j}(f,h,v) = \alpha\sum_{i,j}[(f_{i,j} - f_{i,j-1})^2(1-h_{i,j}) + (f_{i,j} - f_{i-1,j})^2(1-v_{i,j})] + \gamma(h_{i,j} + v_{i,j})$$

$$(3.1)$$

and $\alpha$ and $\gamma$ are positive parameters.

The first term, contains the interaction between the field and the line processes: if the horizontal or vertical gradient is very high at site $(i,j)$ the corresponding line process will be very likely to be active ($h_{i,j} = 1$ or $v_{i,j} = 1$), to make the visual cost function decrease and signal a discontinuity. The second term takes into account the price we pay each time we create a discontinuity and is necessary to prevent the creation of discontinuities everywhere. The effective cost function (2.7) then becomes

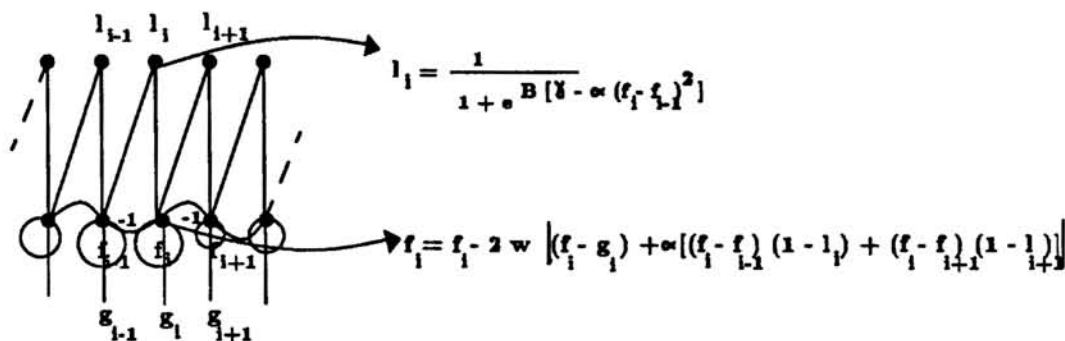

Figure 2: *The network is represented for the one dimensional case. The lines are the connections*

$$V_{ij}^{eff} = \sum_{i,j}\left[\lambda_{ij}(f_{i,j}-g_{i,j})^2+\alpha(\Delta_{i,j}^h)^2+(\Delta_{i,j}^v)^2-\frac{1}{\beta}ln[(1+e^{-\beta(\gamma-\alpha\Delta_{i,j}^h{}^2)})(1+e^{-\beta(\gamma-\alpha\Delta_{i,j}^v{}^2)})]\right]$$

(3.2)

where $\Delta_{i,j}^h = f_{i,j} - f_{i-1,j}$, $\Delta_{i,j}^v = f_{i,j} - f_{i,j-1}$ and (2.5) is then given by

$$\bar{h}_{i,j} = \frac{1}{1+e^{\beta(\gamma-\alpha(\bar{f}_{i,j}-\bar{f}_{i-1,j})^2)}} \quad and \quad \bar{v}_{i,j} = \frac{1}{1+e^{\beta(\gamma-\alpha(\bar{f}_{i,j}-\bar{f}_{i,j-1})^2)}}$$

(3.3).

we point out here that while the line process field is a binary field, its mean value is a continuous (analog) function in the range between 0 and 1.

Discretizing (2.8) in time and applying for (3.2), we obtain

$$\bar{f}_{ij}^{n+1} = \bar{f}_{ij}^n - \omega\left[\lambda_{ij}(\bar{f}_{i,j}^n - g_{i,j}) - \alpha(\bar{f}_{i,j}^n - \bar{f}_{i,j-1}^n)(1-\bar{v}_{i,j}^n) + \alpha(\bar{f}_{i,j+1}^n - \bar{f}_{i,j}^n)(1-\bar{v}_{i,j+1}^n)\right.$$
$$\left. -\alpha(\bar{f}_{i,j}^n - \bar{f}_{i-1,j}^n)(1-\bar{h}_{i,j}^n) + \alpha(\bar{f}_{i+1,j}^n - \bar{f}_{i,j}^n)(1-\bar{h}_{i+1,j}^n)\right]$$

(3.4)

where $\bar{h}_{i,j}$ and $\bar{v}_{i,j}$ are given by the network (3.3) and $n$ is the time step on the algorithm. We notice that (3.4) is coupled with (3.3) such that the field f is updated by (3.4) at step $n$ and then (3.3) updates the field $h$ and $v$ before (3.4) updates field $f$ again at step $n+1$.

This is a simple unsupervised neural network where the imput are the fields $f$ and the output is the line process field $h$ or $v$. This network is coupled to the network (2.8) to solve for the field $f$ and then constitute the global network for this problem (see figure 2). It has been shown by many authors and (Geiger and Yuille, 1989) that these class of networks is equivalent to Hcpfield networks (Hopfield, 1984) (Koch et. al., 1985).

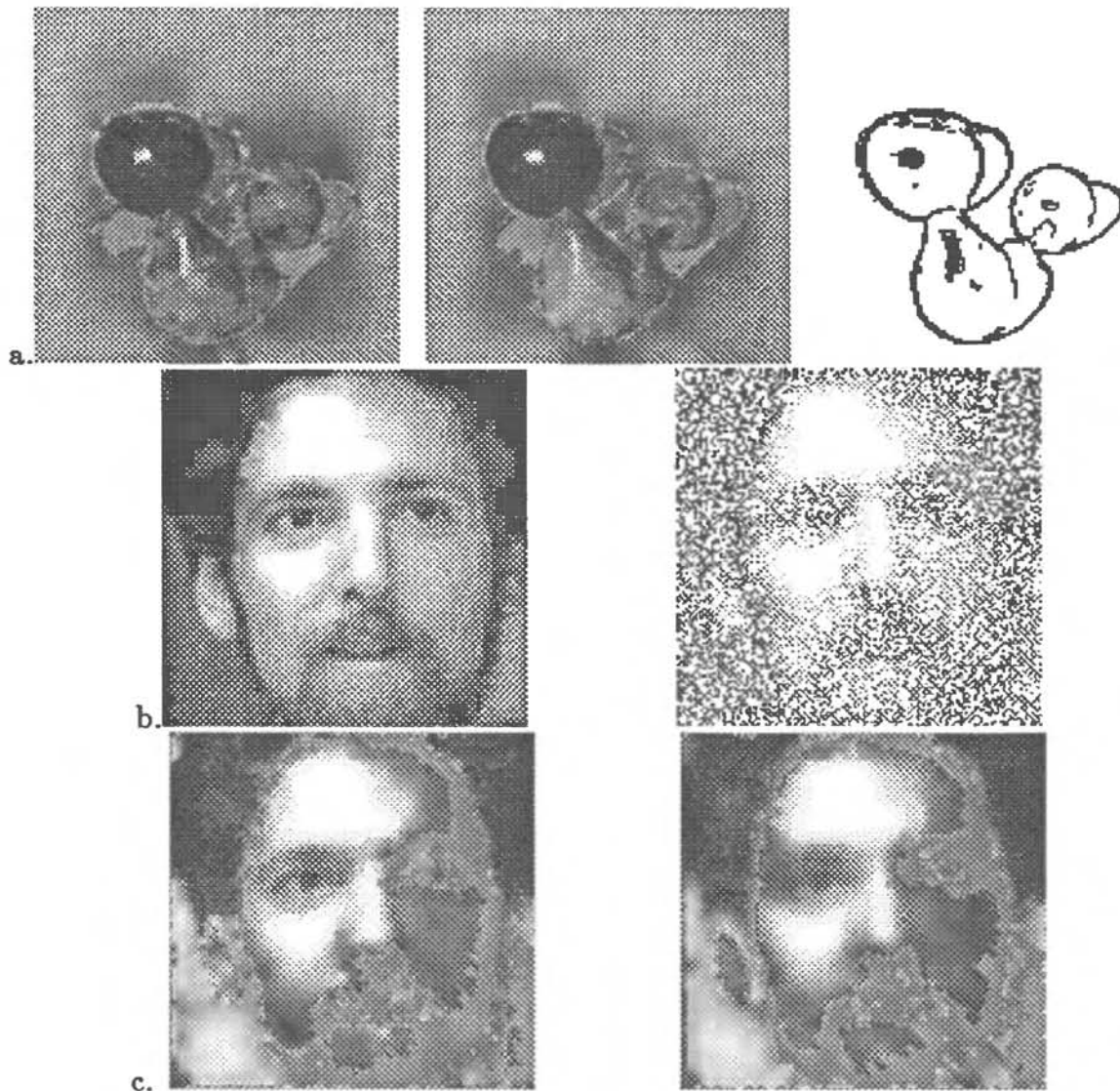

**Figure 3:** *a. The still life image 128 × 128 pixels. The image smoothed with γ = 1400 and α = 4 for 9 iterations. The line process field (needs thinning). b. A face image of 128 × 128 pixels. Randomly chosen 50 % of the original image (for display the other 50% are filled with white dots). c. The network described above is applied to smooth and fill in using the same parameters and for 10 iterations. For comparison we show the results of simply blurring the sparse data (no line process field).*

An important connection we make is to show (Geiger and Girosi, 1989) (Geiger, 1989) that the work of Blake and Zisserman (Blake and Zisserman, 1987) can be seen as an approximation of these results.

In the zero temperature limit ($\beta \to \infty$) (3.3) becomes the Heaviside function (1 or 0) and the interpretation is simple: when the horizontal or vertical gradient are larger than a threshold ($\sqrt{\frac{\gamma}{\alpha}}$) a vertical or horizontal discontinuity is created.

## 4   Results

We applied the network to a real still life image and the result was an enhancement of specular edges, shadow edges and some other contours while smoothing out the noise (see Figure 3a). This result is consistent with all the images we have used. From one face image we produced sparse data by randomly suppressing 50% of the data. (see Figure 3b). We then applied the neural network to reconstruct the image.

### Acknowledgements

We are grateful to Tomaso Poggio for his guidance and support.

## Footnotes

[1] New address is Siemens Corporate Research, 755 College Road East, Princeton NJ 08540

### References

A. Blake and A. Zisserman. (1987) *Visual Reconstruction*. Cambridge, Mass: MIT Press.

E. Gamble and D. Geiger and T. Poggio and D. Weinshall. (1989) *Integration of vision modules and labeling of surface discontinuities*. Invited paper to IEEE Trans. Sustems, Man & Cybernetics, December.

D. Geiger and F. Girosi. (1989) *Parallel and deterministic algorithms for MRFs: surface reconstruction and integration*. A.I. Memo No.1114. Artificial Intelligence Laboratory of MIT.

D. Geiger. (1989) *Visual models with statistical field theory*. Ph.D. thesis. MIT, Physics department and Artificial Intelligence Laboratory.

D. Geiger and A. Yuille. (1989) *A common framework for image segmentation and surface reconstruction*. Harvard Robotics Laboratory Technical Report, 89-7, Harvard, August.

S. Geman and D. Geman. (1984) *Stochastic Relaxation, Gibbs Distributions, and the Bayesian Restoration of Images*. Pattern Analysis and Machine Intelligence, PAMI-6:721–741.

J. J. Hopfield. (1984) *Neurons with graded response have collective computational properties like those of two-state neurons*. Proc. Natl. Acad. Sci., 81:3088-3092,

S. Kirkpatrick and C. D. Gelatt and M. P. Vecchi. (1983) *Optimization by Simulated Annealing*. Science. 220:219–227.

C. Koch and J. Marroquin and A. Yuille. (1985) *Analog 'Neuronal' Networks in Early Vision*. Proc. Natl. Acad. Sci., 83:4263-4267.

J. L. Marroquin and S. Mitter and T. Poggio. (1987) *Probabilistic Solution of Ill-Posed Problems in Computational Vision*. J. Amer. Stat. Assoc., 82:76-89.
